# Efficient Multiscale Sampling from Products of Gaussian Mixtures

**Alexander T. Ihler, Erik B. Sudderth, William T. Freeman, and Alan S. Willsky**
Department of Electrical Engineering and Computer Science
Massachusetts Institute of Technology
*ihler@mit.edu, esuddert@mit.edu, billf@ai.mit.edu, willsky@mit.edu*

## Abstract

The problem of approximating the product of several Gaussian mixture distributions arises in a number of contexts, including the nonparametric belief propagation (NBP) inference algorithm and the training of product of experts models. This paper develops two multiscale algorithms for sampling from a product of Gaussian mixtures, and compares their performance to existing methods. The first is a multiscale variant of previously proposed Monte Carlo techniques, with comparable theoretical guarantees but improved empirical convergence rates. The second makes use of approximate kernel density evaluation methods to construct a fast approximate sampler, which is guaranteed to sample points to within a tunable parameter $\epsilon$ of their true probability. We compare both multiscale samplers on a set of computational examples motivated by NBP, demonstrating significant improvements over existing methods.

## 1 Introduction

Gaussian mixture densities are widely used to model complex, multimodal relationships. Although they are most commonly associated with parameter estimation procedures like the EM algorithm, kernel or Parzen window nonparametric density estimates [1] also take this form for Gaussian kernel functions. Products of Gaussian mixtures naturally arise whenever multiple sources of statistical information, each of which is individually modeled by a mixture density, are combined. For example, given two independent observations $y_1, y_2$ of an unknown variable $x$, the joint likelihood $p(y_1, y_2|x) \propto p(y_1|x)p(y_2|x)$ is equal to the product of the marginal likelihoods. In a recently proposed nonparametric belief propagation (NBP) [2, 3] inference algorithm for graphical models, Gaussian mixture products are the mechanism by which nodes fuse information from different parts of the graph. Product densities also arise in the product of experts (PoE) [4] framework, in which complex densities are modeled as the product of many "local" constraint densities.

The primary difficulty associated with products of Gaussian mixtures is computational. The product of $d$ mixtures of $N$ Gaussians is itself a Gaussian mixture with $N^d$ components. In many practical applications, it is infeasible to explicitly construct these components, and therefore intractable to build a smaller approximating mixture using the EM algorithm. Mixture products are thus typically approximated by drawing samples from the product density. These samples can be used to either form a Monte Carlo estimate of a desired expectation [4], or construct a kernel density estimate approximating the true product [2].

Although exact sampling requires exponential cost, Gibbs sampling algorithms may often be used to produce good approximate samples [2, 4].

When accurate approximations are required, existing methods for sampling from products of Gaussian mixtures often require a large computational cost. In particular, sampling is the primary computational burden for both NBP and PoE. This paper develops a pair of new sampling algorithms which use multiscale, KD-Tree [5] representations to improve accuracy and reduce computation. The first is a multiscale variant of existing Gibbs samplers [2, 4] with improved empirical convergence rate. The second makes use of approximate kernel density evaluation methods [6] to construct a fast $\epsilon$-*exact* sampler which, in contrast with existing methods, is guaranteed to sample points to within a tunable parameter $\epsilon$ of their true probability. Following our presentation of the algorithms, we demonstrate their performance on a set of computational examples motivated by NBP and PoE.

## 2 Products of Gaussian Mixtures

Let $\{p_1(x), \ldots, p_d(x)\}$ denote a set of $d$ mixtures of $N$ Gaussian densities, where

$$p_i(x) = \sum_{l_i} w_{l_i} \mathcal{N}(x; \mu_{l_i}, \Lambda_i) \tag{1}$$

Here, $l_i$ are a set of labels for the $N$ mixture components in $p_i(x)$, $w_{l_i}$ are the normalized component weights, and $\mathcal{N}(x; \mu_{l_i}, \Lambda_i)$ denotes a normalized Gaussian density with mean $\mu_{l_i}$ and diagonal covariance $\Lambda_i$. For simplicity, we assume that all mixtures are of equal size $N$, and that the variances $\Lambda_i$ are uniform within each mixture, although the algorithms which follow may be readily extended to problems where this is not the case. Our goal is to efficiently sample from the $N^d$ component mixture density $p(x) \propto \prod_{i=1}^{d} p_i(x)$.

### 2.1 Exact Sampling

Sampling from the product density can be decomposed into two steps: randomly select one of the product density's $N^d$ components, and then draw a sample from the corresponding Gaussian. Let each product density component be labeled as $L = [l_1, \ldots, l_d]$, where $l_i$ labels one of the $N$ components of $p_i(x)$.[1] The relative weight of component $L$ is given by

$$w_L = \frac{\prod_{i=1}^{d} w_{l_i} \mathcal{N}(x; \mu_{l_i}, \Lambda_i)}{\mathcal{N}(x; \mu_L, \Lambda_L)} \qquad \Lambda_L^{-1} = \sum_{i=1}^{d} \Lambda_i^{-1} \qquad \Lambda_L^{-1} \mu_L = \sum_{i=1}^{d} \Lambda_i^{-1} \mu_{l_i} \tag{2}$$

where $\mu_L, \Lambda_L$ are the mean and variance of product component $L$, and this equation may be evaluated at any $x$ (the value $x = \mu_L$ may be numerically convenient). To form the product density, these weights are normalized by the *weight partition function* $Z \triangleq \sum_L w_L$.

Determining $Z$ exactly takes $\mathcal{O}(N^d)$ time, and given this constant we can draw $N$ samples from the distribution in $\mathcal{O}(N^d)$ time and $\mathcal{O}(N)$ storage. This is done by drawing and sorting $N$ uniform random variables on the interval $[0, 1]$, and then computing the cumulative distribution of $p(L) = w_L/Z$ to determine which, if any, samples are drawn from each $L$.

### 2.2 Importance Sampling

Importance sampling is a Monte Carlo method for approximately sampling from (or computing expectations of) an intractable distribution $p(x)$, using a *proposal distribution* $q(x)$ for which sampling is feasible [7]. To draw $N$ samples from $p(x)$, an importance sampler draws $M \geq N$ samples $x_i \sim q(x)$, and assigns the $i^{th}$ sample weight $w_i \propto p(x_i)/q(x_i)$. The weights are then normalized by $Z = \sum_i w_i$, and $N$ samples are drawn (with replacement) from the discrete distribution $\bar{p}(x_i) = w_i/Z$.

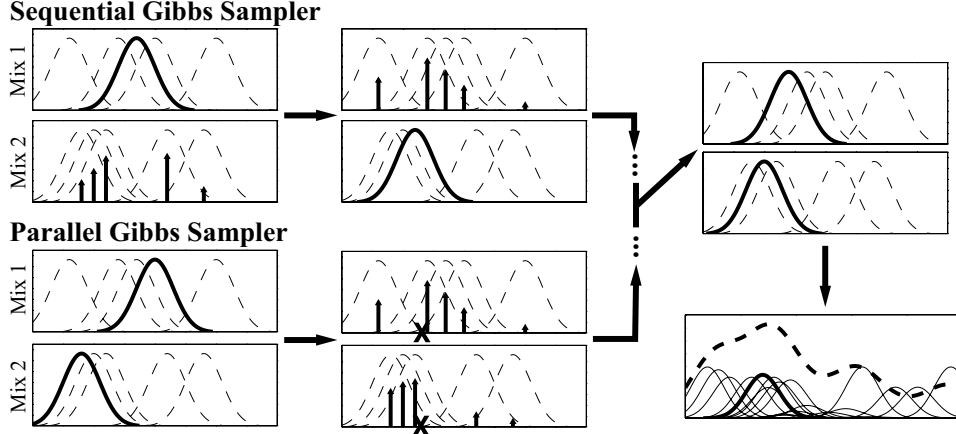

**Sequential Gibbs Sampler**

Mix 1 / Mix 2

**Parallel Gibbs Sampler**

Mix 1 / Mix 2

Figure 1: Two possible Gibbs samplers for a product of 2 mixtures of 5 Gaussians. Arrows show the weights assigned to each label. *Top left:* At each iteration, one label is sampled conditioned on the other density's current label. *Bottom left:* Alternate between sampling a data point $X$ conditioned on the current labels, and resampling all labels in parallel. *Right:* After $\kappa$ iterations, both Gibbs samplers identify mixture labels corresponding to a single kernel (solid) in the product density (dashed).

For products of Gaussian mixtures, we consider two different proposal distributions. The first, which we refer to as *mixture importance sampling*, draws each sample by randomly selecting one of the $d$ input mixtures, and sampling from its $N$ components ($q(x) = p_i(x)$). The remaining $d - 1$ mixtures then provide the importance weight ($w_i = \prod_{j \neq i} p_j(x_i)$). This is similar to the method used to combine density trees in [8]. Alternatively, we can approximate each input mixture $p_i(x)$ by a single Gaussian density $q_i(x)$, and choose $q(x) \propto \prod_i q_i(x)$. We call this procedure *Gaussian importance sampling*.

### 2.3 Gibbs Sampling

Sampling from Gaussian mixture products is difficult because the joint distribution over product density labels, as defined by equation (2), is complicated. However, conditioned on the labels of all but one mixture, we can compute the conditional distribution over the remaining label in $\mathcal{O}(N)$ operations, and easily sample from it. Thus, we may use a Gibbs sampler [9] to draw asymptotically unbiased samples, as illustrated in Figure 1. At each iteration, the labels $\{l_j\}_{j \neq i}$ for $d - 1$ of the input mixtures are fixed, and the $i^{th}$ label is sampled from the corresponding conditional density. The newly chosen $l_i$ is then fixed, and another label is updated. After a fixed number of iterations $\kappa$, a single sample is drawn from the product mixture component identified by the final labels. To draw $N$ samples, the Gibbs sampler requires $\mathcal{O}(d\kappa N^2)$ operations; see [2] for further details.

The previously described *sequential Gibbs sampler* defines an iteration over the labels of the input mixtures. Another possibility uses the fact that, given a data point $\bar{x}$ in the product density space, the $d$ input mixture labels are conditionally independent [4]. Thus, one can define a *parallel Gibbs sampler* which alternates between sampling a data point conditioned on the current input mixture labels, and parallel sampling of the mixture labels given the current data point (see Figure 1). The complexity of this sampler is also $\mathcal{O}(d\kappa N^2)$.

## 3 KD–Trees

A KD-tree is a hierarchical representation of a point set which caches statistics of subsets of the data, thereby making later computations more efficient [5]. KD-trees are typically binary trees constructed by successively splitting the data along cardinal axes, grouping points by spatial location. We use the variable $l$ to denote the label of a leaf node (the index of a single point), and $\mathfrak{l}$ to denote a set of leaf labels summarized at a node of the KD-tree.

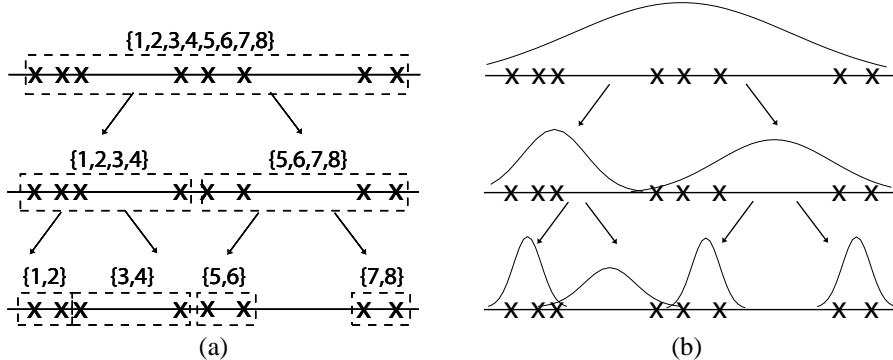

| (a) | (b) |

Figure 2: Two KD-tree representations of the same one-dim. point set. (a) Each node maintains a bounding box (label sets $\mathfrak{l}$ are shown in braces). (b) Each node maintains mean and variance statistics.

Figure 2 illustrates one-dimensional KD-trees which cache different sets of statistics. The first (Figure 2(a)) maintains bounding boxes around the data, allowing efficient computation of distances; similar trees are used in Section 4.2. Also shown in this figure are the label sets $\mathfrak{l}$ for each node. The second (Figure 2(b)) precomputes means and variances of point clusters, providing a multi-scale Gaussian mixture representation used in Section 4.1.

### 3.1 Dual Tree Evaluation

Multiscale representations have been effectively applied to kernel density estimation problems. Given a mixture of $N$ Gaussians with means $\{\mu_i\}$, we would like to evaluate

$$p(x_j) = \sum_i w_i \mathcal{N}(x_j; \mu_i, \Lambda) \qquad (3)$$

at a given set of $M$ points $\{x_j\}$. By representing the means $\{\mu_i\}$ and evaluation points $\{x_j\}$ with two different KD-trees, it is possible to define a *dual–tree* recursion [6] which is much faster than direct evaluation of all $NM$ kernel–point pairs.

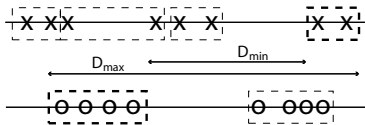

Figure 3: Two KD-tree representations may be combined to efficiently bound the maximum ($D_{\max}$) and minimum ($D_{\min}$) pairwise distances between subsets of the summarized points (bold).

The dual-tree algorithm uses bounding box statistics (as in Figure 2(a)) to approximately evaluate subsets of the data. For any set of labels in the density tree $\mathfrak{l}_\mu$ and location tree $\mathfrak{l}_x$, one may use pairwise distance bounds (see Figure 3) to find upper and lower bounds on

$$\sum_{i \in \mathfrak{l}_\mu} w_i \mathcal{N}(x_j; \mu_i, \Lambda) \quad \text{for any} \quad j \in \mathfrak{l}_x \qquad (4)$$

When the distance bounds are sufficiently tight, the sum in equation (4) may be approximated by a constant, asymptotically allowing evaluation in $\mathcal{O}(N)$ operations [6].

## 4 Sampling using Multiscale Representations

### 4.1 Gibbs Sampling on KD-Trees

Although the pair of Gibbs samplers discussed in Section 2.3 are often effective, they sometimes require a very large number of iterations to produce accurate samples. The most difficult densities are those for which there are multiple widely separated modes, each of which is associated with disjoint subsets of the input mixture labels. In this case, conditioned on a set of labels corresponding to one mode, it is very unlikely that a label or data point corresponding to a different mode will be sampled, leading to slow convergence.

Similar problems have been observed with Gibbs samplers on Markov random fields [9]. In these cases, convergence can often be accelerated by constructing a series of "coarser

scale" approximate models in which the Gibbs sampler can move between modes more easily [10]. The primary challenge in developing these algorithms is to determine procedures for constructing accurate coarse scale approximations. For Gaussian mixture products, KD-trees provide a simple, intuitive, and easily constructed set of coarser scale models.

As in Figure 2(b), each level of the KD-tree stores the mean and variance (biased by kernel size) of the summarized leaf nodes. We start at the same coarse scale for all input mixtures, and perform standard Gibbs sampling on that scale's summary Gaussians. After several iterations, we condition on a data sample (as in the parallel Gibbs sampler of Section 2.3) to infer labels at the next finest scale. Intuitively, by gradually moving from coarse to fine scales, multiscale sampling can better explore all of the product density's important modes.

As the number of sampling iterations approaches infinity, multiscale samplers have the same asymptotic properties as standard Gibbs samplers. Unfortunately, there is no guarantee that multiscale sampling will improve performance. However, our simulation results indicate that it is usually very effective (see Section 5).

## 4.2 Epsilon-Exact Sampling using KD-Trees

In this section, we use KD-trees to efficiently compute an approximation to the partition function $Z$, in a manner similar to the dual tree evaluation algorithm of [6] (see Section 3.1). This leads to an $\epsilon$-*exact* sampler for which a label $L = [l_1, \ldots, l_d]$, with true probability $p_L$, is guaranteed to be sampled with some probability $\hat{p}_L \in [p_L - \epsilon, p_L + \epsilon]$. We denote subsets of labels in the input densities with lowercase script ($\mathfrak{l}_i$), and sets of labels in the product density by $\mathfrak{L} = \mathfrak{l}_1 \times \cdots \times \mathfrak{l}_d$. The approximate sampling procedure is similar to the exact sampler of Section 2.1. We first construct KD-tree representations of each input density (as in Figure 2(a)), and use a *multi–tree* recursion to approximate the partition function $\hat{Z} = \sum \hat{w}_L$ by summarizing sets of labels $\mathfrak{L}$ where possible. Then, we compute the cumulative distribution of the *sets* of labels, giving each label set $\mathfrak{L}$ probability $\hat{w}_{\mathfrak{L}}/\hat{Z}$.

### 4.2.1 Approximate Evaluation of the Weight Partition Function

We first note that the weight function (equation (2)) can be rewritten using terms which involve only pairwise distances (the quotient is computed elementwise):

$$ w_L = \big( \prod_{j=1}^{d} w_{l_j} \big) \cdot \prod_{(l_i, l_j > i)} \mathcal{N}(\mu_{l_i}; \mu_{l_j}, \Lambda_{(i,j)}) \qquad \text{where} \qquad \Lambda_{(i,j)} = \frac{\Lambda_i \Lambda_j}{\Lambda_L} \quad (5) $$

This equation may be divided into two parts: a weight contribution $\prod_{i=1}^{d} w_{l_i}$, and a distance contribution (which we denote by $K_L$) expressed in terms of the pairwise distances between kernel centers. We use the KD-trees' distance bounds to compute bounds on each of these pairwise distance terms for a collection of labels $\mathfrak{L} = \mathfrak{l}_1 \times \cdots \times \mathfrak{l}_d$. The product of the upper (lower) pairwise bounds is itself an upper (lower) bound on the total distance contribution for any label $L$ within the set; denote these bounds by $K_{\mathfrak{L}}^+$ and $K_{\mathfrak{L}}^-$, respectively.[2]

By using the mean $K_{\mathfrak{L}}^* = \frac{1}{2} \big( K_{\mathfrak{L}}^+ + K_{\mathfrak{L}}^- \big)$ to approximate $K_L$, we incur a maximum error $\frac{1}{2} \big( K_{\mathfrak{L}}^+ - K_{\mathfrak{L}}^- \big)$ for any label $L \in \mathfrak{L}$. If this error is less than $Z\delta$ (which we ensure by comparing to a running lower bound $Z_{min}$ on $Z$), we treat it as constant over the set $\mathfrak{L}$ and approximate the contribution to $Z$ by

$$ \sum_{L \in \mathfrak{L}} \hat{w}_L = K_{\mathfrak{L}}^* \sum_{L \in \mathfrak{L}} \big( \prod_i w_{l_i} \big) = K_{\mathfrak{L}}^* \prod_i \big( \sum_{l_i \in \mathfrak{l}_i} w_{l_i} \big) \qquad (6) $$

This is easily calculated using cached statistics of the weight contained in each set. If the error is larger than $Z\delta$, we need to refine at least one of the label sets; we use a heuristic to make this choice. This procedure is summarized in Algorithm 1. Note that all of the

MultiTree($[\mathfrak{l}_1, \ldots, \mathfrak{l}_d]$)

1. For each pair of distributions $(i, j > i)$, use their bounding boxes to compute
   (a) $K_{max}^{(i,j)} \geq \max_{l_i \in \mathfrak{l}_i, l_j \in \mathfrak{l}_j} \mathcal{N}(x_{l_i} - x_{l_j}; 0, \Lambda_{(i,j)})$
   (b) $K_{min}^{(i,j)} \leq \min_{l_i \in \mathfrak{l}_i, l_j \in \mathfrak{l}_j} \mathcal{N}(x_{l_i} - x_{l_j}; 0, \Lambda_{(i,j)})$
2. Find $K_{max} = \prod_{(i,j>i)} K_{max}^{(i,j)}$ and $K_{min} = \prod_{(i,j>i)} K_{min}^{(i,j)}$
3. If $\frac{1}{2}(K_{max} - K_{min}) \leq Z_{min}\delta$, approximate this combination of label sets:
   (a) $\hat{w}_{\mathfrak{L}} = \frac{1}{2}(K_{max} + K_{min})(\prod w_{\mathfrak{l}_i})$, where $w_{\mathfrak{l}_i} = \sum_{l_i \in \mathfrak{l}_i} w_{l_i}$ is cached by the KD-trees
   (b) $Z_{min} = Z_{min} + K_{min}(\prod w_{\mathfrak{l}_i})$
   (c) $\hat{Z} = \hat{Z} + \hat{w}_{\mathfrak{L}}$
4. Otherwise, refine one of the label sets:
   (a) Find $\arg\max_{(i,j)} K_{max}^{(i,j)}/K_{min}^{(i,j)}$ such that range($\mathfrak{l}_i$) $\geq$ range($\mathfrak{l}_j$).
   (b) Call recursively:
      i. MultiTree($[\mathfrak{l}_1, \ldots, \text{Nearer}(\text{Left}(\mathfrak{l}_i), \text{Right}(\mathfrak{l}_i), \mathfrak{l}_j), \ldots, \mathfrak{l}_d]$)
      ii. MultiTree($[\mathfrak{l}_1, \ldots, \text{Farther}(\text{Left}(\mathfrak{l}_i), \text{Right}(\mathfrak{l}_i), \mathfrak{l}_j), \ldots, \mathfrak{l}_d]$)
   where Nearer(Farther) returns the nearer (farther) of the first two arguments to the third.

Algorithm 1: Recursive multi-tree algorithm for approximately evaluating the partition function $Z$ of the product of $d$ Gaussian mixture densities represented by KD–trees. $Z_{min}$ denotes a running lower bound on the partition function, while $\hat{Z}$ is the current estimate. Initialize $Z_{min} = \hat{Z} = 0$.

Given the final partition function estimate $\hat{Z}$, repeat Algorithm 1 with the following modifications:
   3. (c) If $\hat{c} \leq \hat{Z}u_j < \hat{c} + \hat{w}_{\mathfrak{L}}$ for any $j$, draw $L \in \mathfrak{L}$ by sampling $l_i \in \mathfrak{l}_i$ with weight $w_{l_i}/w_{\mathfrak{l}_i}$
   3. (d) $\hat{c} = \hat{c} + \hat{w}_{\mathfrak{L}}$

Algorithm 2: Recursive multi-tree algorithm for approximate sampling. $\hat{c}$ denotes the cumulative sum of weights $\hat{w}_{\mathfrak{L}}$. Initialize by sorting $N$ uniform $[0, 1]$ samples $\{u_j\}$, and set $Z_{min} = \hat{c} = 0$.

quantities required by this algorithm may be stored within the KD–trees, avoiding searches over the sets $\mathfrak{l}_i$. At the algorithm's termination, the total error is bounded by

$$|Z - \hat{Z}| \leq \sum_L |w_L - \hat{w}_L| \leq \sum_L \frac{1}{2}\left(K_{\mathfrak{L}}^+ - K_{\mathfrak{L}}^-\right) \prod w_{l_i} \leq Z\delta \sum_L \prod w_{l_i} \leq Z\delta \quad (7)$$

where the last inequality follows because each input mixture's weights are normalized. This guarantees that our estimate $\hat{Z}$ is within a fractional tolerance $\delta$ of its true value.

### 4.2.2 Approximate Sampling from the Cumulative Distribution

To use the partition function estimate $\hat{Z}$ for approximate sampling, we repeat the approximation process in a manner similar to the exact sampler: draw $N$ sorted uniform random variables, and then locate these samples in the cumulative distribution. We do not explicitly construct the cumulative distribution, but instead use the same approximate partial weight sums used to determine $\hat{Z}$ (see equation (6)) to find the block of labels $\mathfrak{L} = \mathfrak{l}_1 \times \cdots \times \mathfrak{l}_d$ associated with each sample. Since all labels $L \in \mathfrak{L}$ within this block have approximately equal distance contribution $K_L \approx K_{\mathfrak{L}}^*$, we independently sample a label $l_i$ within each set $\mathfrak{l}_i$ proportionally to the weight $w_{l_i}$.

This procedure is shown in Algorithm 2. Note that, to be consistent about when approximations are made and thus produce weights $\hat{w}_{\mathfrak{L}}$ which still sum to $\hat{Z}$, we repeat the procedure for computing $\hat{Z}$ exactly, including recomputing the running lower bound $Z_{min}$. This algorithm is guaranteed to sample each label $L$ with probability $\hat{p}_L \in [p_L - \epsilon, p_L + \epsilon]$:

$$|\hat{p}_L - p_L| = \left|\frac{\hat{w}_L}{\hat{Z}} - \frac{w_L}{Z}\right| \leq \frac{2\delta}{1-\delta} \triangleq \epsilon \quad (8)$$

**Proof:** From our bounds on the error of $K_{\mathfrak{L}}^*$, $|\frac{w_L}{Z} - \frac{\hat{w}_L}{Z}| = \frac{|K_L - K_L^*|}{Z}\prod w_{l_i} \leq \delta(\prod w_{l_i}) \leq \delta$ and $|\frac{\hat{w}_L}{Z} - \frac{\hat{w}_L}{\hat{Z}}| = \frac{\hat{w}_L}{Z}|1 - \frac{1}{\hat{Z}/Z}| \leq \frac{\hat{w}_L}{Z}|1 - \frac{1}{1-\delta}| \leq \frac{\hat{w}_L}{Z}\frac{\delta}{1-\delta} \leq \frac{1+\delta}{1-\delta}\delta$. Thus, the estimated probability of choosing label $L$ has at most error $|\frac{w_L}{Z} - \frac{\hat{w}_L}{\hat{Z}}| \leq |\frac{w_L}{Z} - \frac{\hat{w}_L}{Z}| + |\frac{\hat{w}_L}{Z} - \frac{\hat{w}_L}{\hat{Z}}| \leq \frac{2\delta}{1-\delta}$. $\square$

# 5 Computational Examples

## 5.1 Products of One–Dimensional Gaussian Mixtures

In this section, we compare the sampling methods discussed in this paper on three challenging one–dimensional examples, each involving products of mixtures of 100 Gaussians (see Figure 4). We measure performance by drawing 100 samples, constructing a kernel density estimate using likelihood cross–validation [1], and calculating the KL divergence from the true product density. We repeat this test 250 times for each of a range of parameter settings of each algorithm, and plot the average KL divergence versus computation time.

For the product of three mixtures in Figure 4(a), the multiscale (MS) Gibbs samplers dramatically outperform standard Gibbs sampling. In addition, we see that sequential Gibbs sampling is more accurate than parallel. Both of these differences can be attributed to the bimodal product density. However, the most effective algorithm is the $\epsilon$–exact sampler, which matches exact sampling's performance in far less time (0.05 versus 2.75 seconds). For a product of five densities (Figure 4(b)), the cost of exact sampling increases to 7.6 hours, but the $\epsilon$–exact sampler matches its performance in less than one minute. Even faster, however, is the sequential MS Gibbs sampler, which takes only 0.3 seconds.

For the previous two examples, mixture importance sampling (IS) is nearly as accurate as the best multiscale methods (Gaussian IS seems ineffective). However, in cases where all of the input densities have little overlap with the product density, mixture IS performs very poorly (see Figure 4(c)). In contrast, multiscale samplers perform very well in such situations, because they can discard large numbers of low weight product density kernels.

## 5.2 Tracking an Object using Nonparametric Belief Propagation

NBP [2] solves inference problems on non–Gaussian graphical models by propagating the results of local sampling computations. Using our multiscale samplers, we applied NBP to a simple tracking problem in which we observe a slowly moving object in a sea of randomly shifting clutter. Figure 5 compares the posterior distributions of different samplers two time steps after an observation containing only clutter. $\epsilon$–exact sampling matches the performance of exact sampling, but takes half as long. In contrast, a standard particle filter [7], allowed ten times more computation, loses track. As in the previous section, multiscale Gibbs sampling is much more accurate than standard Gibbs sampling.

# 6 Discussion

For products of a few mixtures, the $\epsilon$–exact sampler is extremely fast, and is guaranteed to give good performance. As the number of mixtures grow, $\epsilon$–exact sampling may become overly costly, but the sequential multiscale Gibbs sampler typically produces accurate samples with only a few iterations. We are currently investigating the performance of these algorithms on large–scale nonparametric belief propagation applications.

## Footnotes

[1]Throughout this paper, we use lowercase letters ($l_i$) to label input density components, and capital letters ($L = [l_1, \ldots, l_d]$) to label the corresponding product density components.

[2]We can also use multipole methods such as the Fast Gauss Transform [11] to efficiently compute alternate, potentially tighter bounds on the pairwise values.

# References

[1] B. W. Silverman. *Density Estimation for Statistics and Data Analysis*. Chapman & Hall, 1986.

[2] E. B. Sudderth, A. T. Ihler, W. T. Freeman, and A. S. Willsky. Nonparametric belief propagation. In *CVPR*, 2003.

[3] M. Isard. PAMPAS: Real–valued graphical models for computer vision. In *CVPR*, 2003.

[4] G. E. Hinton. Training products of experts by minimizing contrastive divergence. Technical Report 2000-004, Gatsby Computational Neuroscience Unit, 2000.

[5] K. Deng and A. W. Moore. Multiresolution instance-based learning. In *IJCAI*, 1995.

[6] A. G. Gray and A. W. Moore. Very fast multivariate kernel density estimation. In *JSM*, 2003.

[7] A. Doucet, N. de Freitas, and N. Gordon, editors. *Sequential Monte Carlo Methods in Practice*. Springer-Verlag, New York, 2001.

[8] S. Thrun, J. Langford, and D. Fox. Monte Carlo HMMs. In *ICML*, pages 415–424, 1999.

[9] S. Geman and D. Geman. Stochastic relaxation, Gibbs distributions, and the Bayesian restoration of images. *IEEE Trans. PAMI*, 6(6):721–741, November 1984.

[10] J. S. Liu and C. Sabatti. Generalised Gibbs sampler and multigrid Monte Carlo for Bayesian computation. *Biometrika*, 87(2):353–369, 2000.

[11] J. Strain. The fast Gauss transform with variable scales. *SIAM J. SSC*, 12(5):1131–1139, 1991.

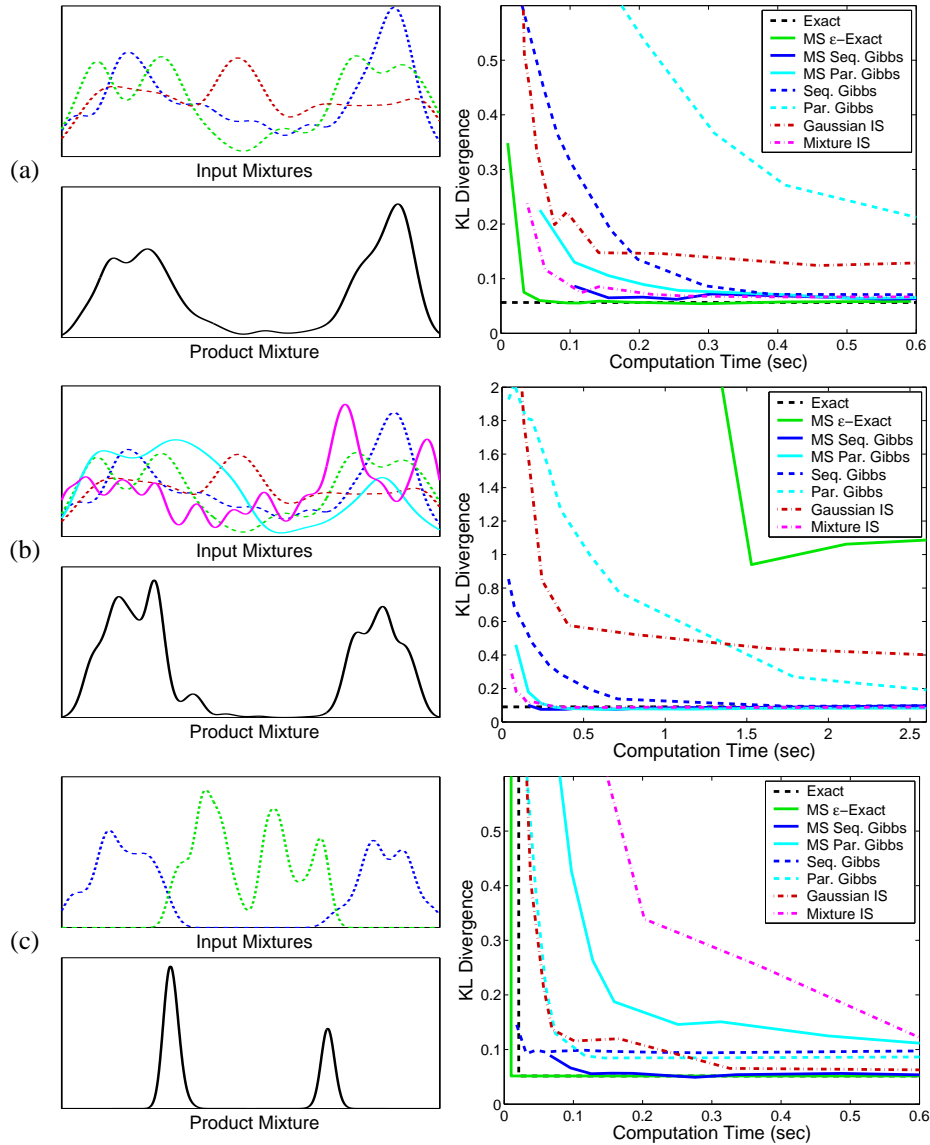

Figure 4: Comparison of average sampling accuracy versus computation time for different algorithms (see text). (a) Product of 3 mixtures (exact requires 2.75 sec). (b) Product of 5 mixtures (exact requires 7.6 hours). (c) Product of 2 mixtures (exact requires 0.02 sec).

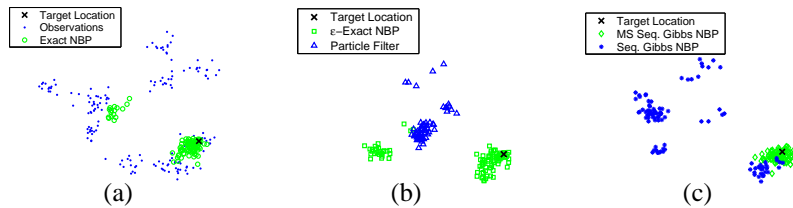

Figure 5: Object tracking using NBP. Plots show the posterior distributions two time steps after an observation containing only clutter. The particle filter and Gibbs samplers are allowed equal computation. (a) Latest observations, and exact sampling posterior. (b) $\epsilon$–exact sampling is very accurate, while a particle filter loses track. (c) Multiscale Gibbs sampling leads to improved performance.